# An Online Algorithm for
# Large Scale Image Similarity Learning

**Gal Chechik**
Google
Mountain View, CA
gal@google.com

**Varun Sharma**
Google
Bengalooru, Karnataka, India
vasharma@google.com

**Uri Shalit**
ICNC, The Hebrew University
Israel
uri.shalit@mail.huji.ac.il

**Samy Bengio**
Google
Mountain View, CA
bengio@google.com

## Abstract

Learning a measure of similarity between pairs of objects is a fundamental problem in machine learning. It stands in the core of classification methods like kernel machines, and is particularly useful for applications like searching for images that are similar to a given image or finding videos that are relevant to a given video. In these tasks, users look for objects that are not only visually similar but also semantically related to a given object. Unfortunately, current approaches for learning similarity do not scale to large datasets, especially when imposing metric constraints on the learned similarity. We describe OASIS, a method for learning pairwise similarity that is fast and scales linearly with the number of objects and the number of non-zero features. Scalability is achieved through online learning of a bilinear model over sparse representations using a large margin criterion and an efficient hinge loss cost. OASIS is accurate at a wide range of scales: on a standard benchmark with thousands of images, it is more precise than state-of-the-art methods, and faster by orders of magnitude. On 2.7 million images collected from the web, OASIS can be trained within 3 days on a single CPU. The non-metric similarities learned by OASIS can be transformed into metric similarities, achieving higher precisions than similarities that are learned as metrics in the first place. This suggests an approach for learning a metric from data that is larger by orders of magnitude than was handled before.

## 1   Introduction

Learning a pairwise similarity measure from data is a fundamental task in machine learning. Pair distances underlie classification methods like nearest neighbors and kernel machines, and similarity learning has important applications for "query-by-example" in information retrieval. For instance, a user may wish to find images that are similar to (but not identical copies of) an image she has; a user watching an online video may wish to find additional videos about the same subject. In all these cases, we are interested in finding a semantically-related sample, based on the visual content of an image, in an enormous search space. Learning a relatedness function from examples could be a useful tool for such tasks.

A large number of previous studies of learning similarities have focused on metric learning, like in the case of a positive semidefinite matrix that defines a Mahalanobis distance [19]. However, similarity learning algorithms are often evaluated in a context of ranking [16, 5]. When the amount

of training data available is very small, adding positivity constraints for enforcing metric properties is useful for reducing overfitting and improving generalization. However, when sufficient data is available, as in many modern applications, adding positive semi-definitiveness constraints is very costly, and its benefit in terms of generalization may be limited. With this view, we take here an approach that avoids imposing positivity or symmetry constraints on the learned similarity measure.

Some similarity learning algorithms assume that the available training data contains real-valued pairwise similarities or distances. Here we focus on a weaker supervision signal: the *relative* similarity of different pairs [4]. This signal is also easier to obtain, here we extract similarity information from pairs of images that share a common label or are retrieved in response to a common text query in an image search engine.

The current paper presents an approach for learning semantic similarity that scales up to two orders of magnitude larger than current published approaches. Three components are combined to make this approach fast and scalable: First, our approach uses an unconstrained bilinear similarity. Given two images $\mathbf{p_1}$ and $\mathbf{p_2}$ we measure similarity through a bilinear form $\mathbf{p_1 W p_2}$, where the matrix $\mathbf{W}$ is not required to be positive, or even symmetric. Second we use a sparse representation of the images, which allows to compute similarities very fast. Finally, the training algorithm that we developed, OASIS, *Online Algorithm for Scalable Image Similarity learning*, is an online dual approach based on the passive-aggressive algorithm [2]. It minimizes a large margin target function based on the hinge loss, and converges to high quality similarity measures after being presented with a small fraction of the training pairs.

We find that OASIS is both fast and accurate at a wide range of scales: for a standard benchmark with thousands of images, it achieves better or comparable results than existing state-of-the-art methods, with computation times that are shorter by an order of magnitude. For web-scale datasets, OASIS can be trained on more than two million images within three days on a single CPU. On this large scale dataset, human evaluations of OASIS learned similarity show that 35% of the ten nearest neighbors of a given image are semantically relevant to that image.

## 2  Learning Relative Similarity

We consider the problem of learning a pairwise similarity function $S$, given supervision on the *relative similarity* between two pairs of images. The algorithm is designed to scale well with the number of samples and the number of features, by using fast online updates and a sparse representation.

Formally, we are given a set of images $\mathcal{P}$, where each image is represented as a vector $p \in \mathbb{R}^d$. We assume that we have access to an oracle that, given a *query image* $p_i \in \mathcal{P}$, can locate two other images, $p_i^+ \in \mathcal{P}$ and $p_i^- \in \mathcal{P}$, such that $p_i^+ \in \mathcal{P}$ is more relevant to $p_i \in \mathcal{P}$ than $p_i^- \in \mathcal{P}$. Formally, we could write that $relevance(p_i, p_i^+) > relevance(p_i, p_i^-)$. However, unlike methods that assume that a numerical value of the similarity is available, $relevance(p_i, p_j) \in \mathbb{R}$, we use this weaker form of supervision, and only assume that some pairs of images can be ranked by their relevance to a query image $p_i$. The relevance measure could reflect that the relevant image $p_i^+$ belongs to the same class of images as the query image, or reflect any other semantic property of the images.

Our goal is to learn a similarity function $S_W(p_i, p_j)$ parameterized by $\mathbf{W}$ that assigns higher similarity scores to the pairs of more relevant images (with a safety margin),

$$S(p_i, p_i^+) > S(p_i, p_i^-) + 1 , \quad \forall p_i, p_i^+, p_i^- \in \mathcal{P} \quad . \tag{1}$$

In this paper, we consider a parametric similarity function that has a bi-linear form,

$$S_{\mathbf{W}}(p_i, p_j) \equiv p_i^T \, \mathbf{W} \, p_j \tag{2}$$

with $\mathbf{W} \in \mathbb{R}^{d \times d}$. Importantly, if the image vectors $p_i \in \mathbb{R}^d$ are sparse, namely, the number of non-zero entries $k_i \equiv \|p_i\|_0$ is small, $k_i \ll d$, then the value of the score defined in Eq. (2) can be computed very efficiently even when $d$ is large. Specifically, $S_{\mathbf{W}}$ can be computed with complexity of $O(k_i k_j)$ regardless of the dimensionality $d$. To learn a scoring function that obeys the constraints in Eq. (1), we define a global loss $L_{\mathbf{W}}$ that accumulates hinge losses over all possible triplets in the training set: $L_{\mathbf{W}} \equiv \sum_{(p_i, p_i^+, p_i^-) \in \mathcal{P}^3} l_{\mathbf{W}}(p_i, p_i^+, p_i^-)$, with the loss for a single triplet being $l_{\mathbf{W}}(p_i, p_i^+, p_i^-) \equiv \max\left(0, 1 - S_{\mathbf{W}}(p_i, p_i^+) + S_{\mathbf{W}}(p_i, p_i^-)\right)$.

To minimize the global loss $L_{\mathbf{W}}$, we propose an algorithm that is based on the Passive-Aggressive family of algorithms [2]. First, $\mathbf{W}$ is initialized to the identity matrix $\mathbf{W}^0 = I_{d \times d}$. Then, the algorithm iteratively draws a random triplet $(p_i, p_i^+, p_i^-)$, and solves the following convex problem with a soft margin:

$$\mathbf{W}^i = \underset{\mathbf{W}}{\text{argmin}} \quad \frac{1}{2} \|\mathbf{W} - \mathbf{W}^{i-1}\|_{Fro}^2 + C\xi \quad \text{s.t.} \quad l_{\mathbf{W}}(p_i, p_i^+, p_i^-) \leq \xi \quad \text{and} \quad \xi \geq 0 \quad (3)$$

where $\|\cdot\|_{Fro}$ is the Frobenius norm (point-wise $L_2$ norm). At the $i^{th}$ iteration, $\mathbf{W}^i$ is updated to optimize a trade-off between staying close to the previous parameters $\mathbf{W}^{i-1}$ and minimizing the loss on the current triplet $l_{\mathbf{W}}(p_i, p_i^+, p_i^-)$. The *aggressiveness* parameter $C$ controls this trade-off.

To solve the problem in Eq. (3) we follow the derivation in [2]. When $l_{\mathbf{W}}(p_i, p_i^+, p_i^-) = 0$, it is clear that $\mathbf{W}^i = \mathbf{W}^{i-1}$ satisfies Eq. (3) directly. Otherwise, we define the Lagrangian

$$\mathcal{L}(\mathbf{W}, \tau, \xi, \lambda) = \frac{1}{2}\|\mathbf{W} - \mathbf{W}^{i-1}\|_{Fro}^2 + C\xi + \tau(1 - \xi - p_i^T \mathbf{W}(p_i^+ - p_i^-)) - \lambda\xi \qquad (4)$$

where $\tau \geq 0$ and $\lambda \geq 0$ are the Lagrange multipliers. The optimal solution is obtained when the gradient vanishes $\frac{\partial \mathcal{L}(\mathbf{W}, \tau, \xi, \lambda)}{\partial \mathbf{W}} = \mathbf{W} - \mathbf{W}^{i-1} - \tau \mathbf{V}_i = 0$, where $\mathbf{V}_i$ is the gradient matrix at the current step $\mathbf{V}_i = \frac{\partial l_{\mathbf{W}}}{\partial \mathbf{W}} = [p_i^1(p_i^+ - p_i^-), \ldots, p_i^d(p_i^+ - p_i^-)]^T$. When image vectors are sparse, the gradient $\mathbf{V}_i$ is also sparse, hence the update step costs only $O(|p_i|_0 \times (\|p_i^+\|_0 + \|p_i^-\|_0))$, where the $L_0$ norm $\|x\|_0$ is the number of nonzero values in $x$. Differentiating the Lagrangian with respect to $\xi$ we obtain $\frac{\partial \mathcal{L}(\mathbf{W}, \tau, \xi, \lambda)}{\partial \xi} = C - \tau - \lambda = 0$ which, knowing that $\lambda \geq 0$, means that $\tau \leq C$. Plugging back into the Lagrangian in Eq. (4), we obtain $\mathcal{L}(\tau) = -\frac{1}{2}\tau^2\|\mathbf{V}_i\|^2 + \tau(1 - p_i^T \mathbf{W}^{i-1}(p_i^+ - p_i^-))$. Finally, taking the derivative of this second Lagrangian with respect to $\tau$ and using $\tau \leq C$, we obtain

$$\begin{aligned}
\mathbf{W} &= \mathbf{W}^{i-1} + \tau \mathbf{V}_i \\
\tau &= \min\left\{ C, \frac{l_{\mathbf{W}^{i-1}}(p_i, p_i^+, p_i^-)}{\|\mathbf{V}_i\|^2} \right\} .
\end{aligned} \qquad (5)$$

The optimal update for the new $\mathbf{W}$ therefore has a form of a gradient descent step with a step size $\tau$ that can be computed exactly. Applying this algorithm for classification tasks was shown to yield a small cumulative online loss, and selecting the best $\mathbf{W}_i$ during training using a hold-out validation set was shown to achieve good generalization [2].

It should be emphasized that OASIS is not guaranteed to learn a parameter matrix that is positive, or even symmetric. We study variants of OASIS that enforce symmetry or positivity in Sec. 4.3.2.

## 3 Related Work

Learning similarity using relative relevance has been intensively studied, and a few recent approaches aim to address learning at large scale. For small-scale data, there are two main groups of similarity learning approaches. The first approach, learning Mahalanobis distances, can be viewed as learning a linear projection of the data into another space (often of lower dimensionality), where a Euclidean distance is defined among pairs of objects. Such approaches include Fisher's Linear Discriminant Analysis (LDA), relevant component analysis (RCA) [1], supervised global metric learning [18], large margin nearest neighbor (LMNN) [16], and metric learning by collapsing classes [5] (MLCC). Other constraints like sparseness are sometimes induced over the learned metric [14]. See also a review in [19] for more details.

The second family of approaches, learning kernels, is used to improve performance of kernel based classifiers. Learning a full kernel matrix in a non parametric way is prohibitive except for very small data sets. As an alternative, several studies suggested learning a weighted sum of pre-defined kernels [11] where the weights are learned from data. In some applications this was shown to be inferior to uniform weighting of the kernels [12]. The work in [4] further learns a weighting over local distance functions for every image in the training set. Non linear image similarity learning was also studied in the context of dimensionality reduction, as in [8].

Finally, Jain et al [9] (based on Davis et al [3]) aim to learn metrics in an online setting. This work is one of the closest work with respect to OASIS: it learns online a linear model of a [dis-]similarity

| Query image | Top 5 relevant images retrieved by OASIS |
|---|---|

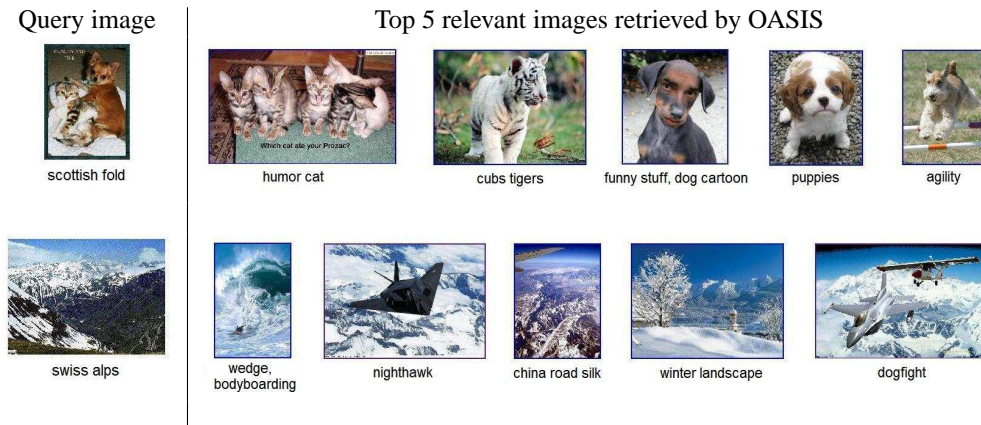

Table 1: OASIS: Successful cases from the web dataset. The relevant text queries for each image are shown beneath the image (not used in training).

function between documents (images); the main difference is that Jain et al [9] try to learn a true distance, imposing positive definiteness constraints, which makes the algorithm more complex and more constrained. We argue in this paper that in the large scale regime, imposing these constraints throughout could be detrimental.

Learning a semantic similarity function between images was also studied in [13]. There, semantic similarity is learned by representing each image by the posterior probability distribution over a predefined set of semantic tags, and then computing the distance between two images as the distance between the two underlying posterior distributions. The representation size of each image therefore grows with the number of semantic classes.

## 4    Experiments

We tested OASIS on two datasets spanning a wide regime of scales. First, we tested its scalability on 2.7 million images collected from the web. Then, to quantitatively compare the precision of OASIS with other, small-scale metric-learning methods, we tested OASIS using *Caltech-256*, a standard machine vision benchmark.

**Image representation**. We use a sparse representation based on *bags of visual words* [6]. These features were systematically tested and found to outperform other features in related tasks, but the details of the visual representation is outside the focus of this paper. Broadly speaking, features are extracted by dividing each image into overlapping square blocks, representing each block by edge and color histograms, and finding the nearest block in a predefined set (dictionary) of $d = 10,000$ vectors of such features. An image is thus represented as the number of times each dictionary visual word was present in it, yielding vectors in $\mathbb{R}^d$ with an average of 70 non-zero values.

**Evaluation protocol**. We evaluated the performance of all algorithms using precision-at-top-$k$, a standard ranking precision measure based on nearest neighbors. For each query image in the test set, all other test images were ranked according to their similarity to the query image, and the number of same-class images among the top $k$ images (the $k$ nearest neighbors) is computed, and then averaged across test images. We also calculated the *mean average precision* (mAP), a measure that is widely used in the information retrieval community.

### 4.1    Web-Scale Experiment

We first tested OASIS on a set of 2.7 million images scraped from the Google image search engine. We collected a set of ∼150K anonymized text queries, and for each of these queries, we had access to a set of relevant images. To compute an image-image relevance measure, we first obtained measures of relevance between images and text queries. This was achieved by collecting anonymized clicks over images collected from the set of text queries. We used this query-image click counts

$C$(query,image) to compute the (unnormalized) probability that two images are co-queried as Relevance(image,image) = $C^T C$. The relevance matrix was then thresholded to keep only the top 1 percent values. We trained OASIS on a training set of 2.3 million images, and tested performance on 0.4 million images. The number of training iterations (each corresponding to sampling one triplet) was selected using a second validation set of around 20000 images, over which the performance saturated after 160 million iterations. Overall, training took a total of ~4000 minutes on a single CPU of a standard modern machine.

Table 1 shows the top five images as ranked by OASIS on two examples of query-images in the test set. In these examples, OASIS captures similarity that goes beyond visual appearance: most top ranked images are about the same concept as the query image, even though that concept was never provided in a textual form, and is inferred in the viewers mind ("dog", "snow"). This shows that learning similarity across co-queried images can indeed capture the semantics of queries even if the queries are not explicitly used during training.

To obtain a quantitative evaluation of the ranking obtained by OASIS we created an evaluation benchmark, by asking human evaluators to mark if a set of candidate images were *semantically relevant* to a set of 25 popular image queries. For each query image, evaluators were presented with the top-10 images ranked by OASIS, mixed with 10 random images. Given the relevance ranking from 30 evaluators, we computed the precision of each OASIS rank as the fraction of people that marked each image as relevant to the query image. On average across all queries and evaluators, OASIS rankings yielded precision of $\sim 40\%$ at the top 10 ranked images.

As an estimate of an "upper bound" on the difficulty of the task, we also computed the precision obtained by human evaluators: For every evaluator, we used the rankings of all other evaluators as ground truth, to compute his precision. As with the ranks of OASIS, we computed the fraction of evaluators that marked an image as relevant, and repeated this separately for every query and human evaluator, providing a measure of "coherence" per query. Fig. 1(a) shows the mean precision obtained by OASIS and human evaluators for every query in our data. For some queries OASIS achieves precision that is very close to that of the mean human evaluator. In many cases OASIS achieves precision that is as good or better than some evaluators.

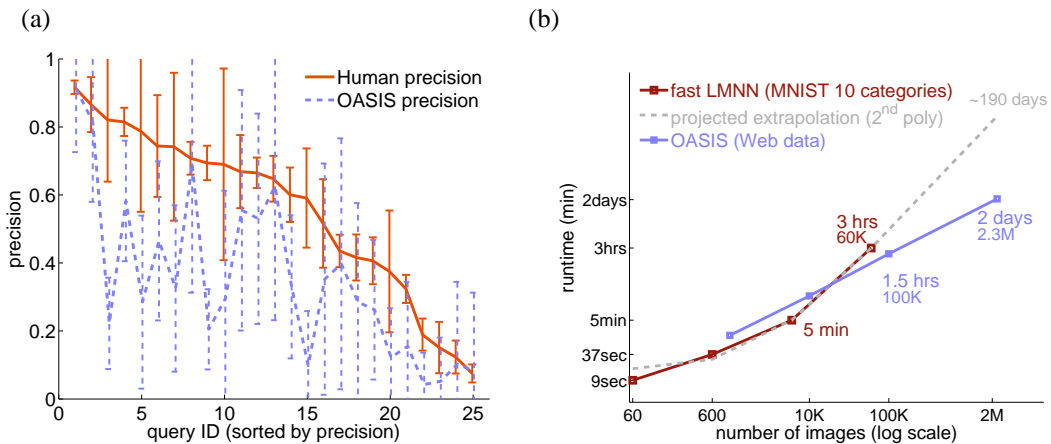

Figure 1: **(a)** Precision of OASIS and human evaluators, per query, using rankings of all (remaining) human evaluators as a ground truth. **(b)** Comparison of the runtime of OASIS and fast-LMNN[17], over a wide range of scales. LMNN results (on MNIST data) are faster than OASIS results on subsets of the web data. However LMNN scales quadratically with the number of samples, hence is three times slower on 60K images, and may be infeasible for handling 2.3 million images.

We further studied how the runtime of OASIS scales with the size of the training set. Figure 1(b) shows that the runtime of OASIS, as found by early stopping on a separate validation set, grows linearly with the train set size. We compare this to the fastest result we found in the literature, based on a fast implementation of LMNN [17]. The LMNN algorithm scales quadratically with the number of objects, although their experiments with MNIST data show that the active set of constraints grows linearly. This could be because MNIST has 10 classes only.

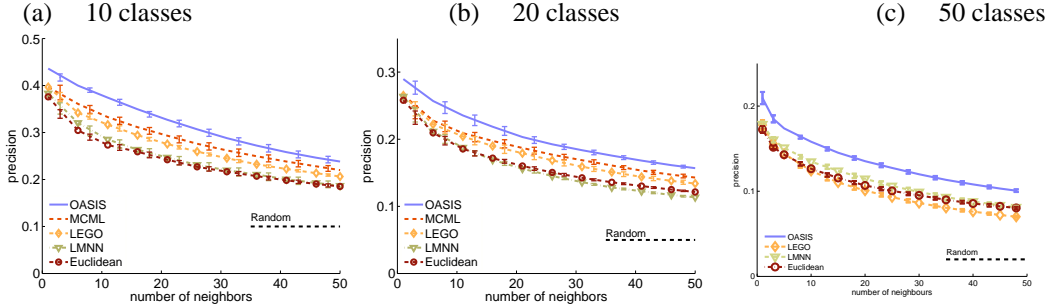

Figure 2: Comparison of the performance of OASIS, LMNN, MCML, LEGO and the Euclidean metric in feature space. Each curve shows the precision at top $k$ as a function of $k$ neighbors. The results are averaged across 5 train/test partitions (40 training images, 25 test images per class), error bars are standard error of the means (s.e.m.), black dashed line denotes chance performance.

## 4.2   Caltech256 Dataset

To compare OASIS with small-scale methods we used the *Caltech256* dataset [7], containing images collected from Google image search and from *PicSearch.com*. Images were assigned to 257 categories and evaluated by humans in order to ensure image quality and relevance. After we have pre-processed the images, and filtered images that were too small, we were left with 29461 images in 256 categories. To allow comparisons with methods that were not optimized for sparse representation, we also reduced the block vocabulary size $d$ from 10000 to 1000.

We compared OASIS with the following metric learning methods.

**(1) Euclidean** - The standard Euclidean distance in feature space (equivalent to using the identity matrix $\mathbf{W} = I_{d \times d}$). **(2) MCML** [5] - Learning a Mahalanobis distance such that same-class samples are mapped to the same point, formulated as a convex problem. **(3) LMNN** [16] - learning a Mahalanobis distance for aiming to have the $k$-nearest neighbors of a given sample belong to the same class while separating different-class samples by a large margin. As a preprocessing phase, images were projected to a basis of the principal components (PCA) of the data, with no dimensionality reduction. **(4) LEGO** [9] - Online learning of a Mahalanobis distance using a Log-Det regularization per instance loss, that is guaranteed to yield a positive semidefinite matrix. We used a variant of LEGO that, like OASIS, learns from relative distances.[1]

We tested all methods on subsets of classes taken from the Caltech256 repository. For OASIS, images from the same class were treated as similar. Each subset was built such that it included semantically diverse categories, controlled for classification difficulty. We tested sets containing 10, 20 and 50 classes, each spanning the range of difficulties.

We used two levels of 5-fold cross validation, one to train the model, and a second to select hyper parameters of each method (early stopping time for OASIS; the $\omega$ parameter for LMNN ($\omega \in \{0.125, 0.25, 0.5\}$), and the regularization parameter $\eta$ for LEGO ($\eta \in \{0.02, 0.08, 0.32\}$). Results reported below were obtained by selecting the best value of the hyper parameter and then training again on the full training set (40 images per class).

Figure 2 compares the precision obtained with OASIS, with the four competing approaches. OASIS achieved consistently superior results throughout the full range of $k$ (number of neighbors) tested, and on all four sets studied. LMNN performance on the *training set* was often high, suggesting that it overfits the training set, as was also observed sometimes by [16].

Table 2 shows the total CPU time in minutes for training all algorithms compared, and for four subsets of classes at sizes 10, 20, 50 and 249. Data is not given when runtime was longer than 5 days or performance was worse than the Euclidean baseline. For the purpose of a fair comparison, we tested two implementations of OASIS: The first was fully implemented Matlab. The second had the core loop of the algorithm implemented in C and called from Matlab. All other methods used

Table 2: Runtime (minutes) on a standard CPU of all compared methods

| num classes | OASIS Matlab | OASIS Matlab+C | MCML Matlab+C | LEGO Matlab | LMNN Matlab+C | fastLMNN Matlab+C |
|---|---|---|---|---|---|---|
| 10 | $42 \pm 15$ | $0.12 \pm .03$ | $1835 \pm 210$ | $143 \pm 44$ | $337 \pm 169$ | $247 \pm 209$ |
| 20 | $45 \pm 8$ | $0.15 \pm .02$ | $7425 \pm 106$ | $533 \pm 49$ | $631 \pm 40$ | $365 \pm 62$ |
| 50 | $25 \pm 2$ | $1.60 \pm .04$ | | $711 \pm 28$ | $960 \pm 80$ | $2109 \pm 67$ |
| 249 | $485 \pm 113$ | $1.13 \pm .15$ | | | | |

code supplied by the authors implemented in Matlab, with core parts implemented in C. Due to compatibility issues, fast-LMNN was run on a different machine, and the given times are rescaled to the same time scale as all other algorithms. LEGO is fully implemented in Matlab. All other code was compiled (mex) to C. The C implementation of OASIS is significantly faster, since Matlab does not use the potential speedup gained by sparse images.

OASIS is significantly faster, with a runtime that is shorter by orders of magnitudes than MCML even on small sets, and about one order of magnitude faster than LMNN. The run time of OASIS and LEGO was measured until the point of early stopping. OASIS memory requirements grow quadratically with the size of the dictionary. For a large dictionary of $10K$, the parameters matrix takes $100M$ floats, or $0.4$ Giga bytes of memory.

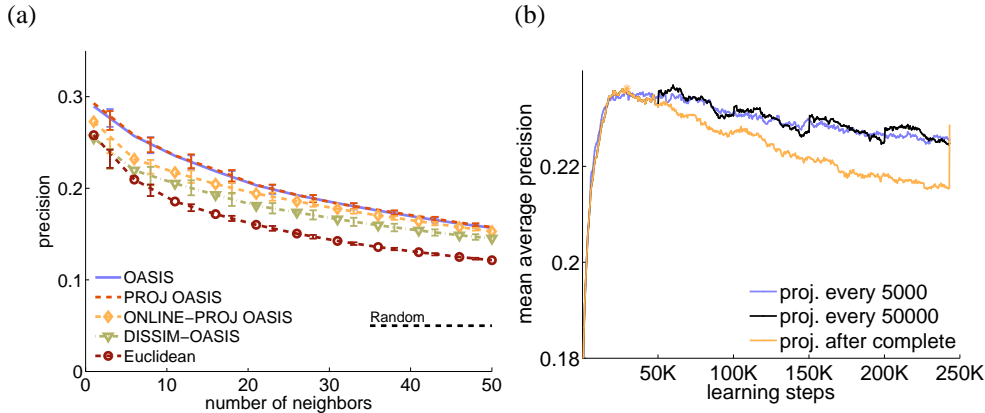

Figure 3: **(a)** Comparing symmetric variants of OASIS on the 20-class subset, similar results obtained with other sets. **(b)** mAP along training for three PSD projection schemes.

## 4.3 Symmetry and positivity

The similarity matrix $\mathbf{W}$ learned by OASIS is not guaranteed to be positive or even symmetric. Some applications, like ranking images by semantic relevance to a given image query are known to be non-symmetric when based on human judgement [15]. However, in some applications symmetry or positivity constraints reflects a prior knowledge that may help in avoiding overfitting. We now discuss variants of OASIS that learn a symmetric or positive matrices.

### 4.3.1 Symmetric similarities

A simple approach to enforce symmetry is to project the OASIS model $\mathbf{W}$ onto the set of symmetric matrices $\mathbf{W}' = sym(\mathbf{W}) = \frac{1}{2}\left(\mathbf{W}^T + \mathbf{W}\right)$. Projection can be done after each update (denoted *Online-Proj-Oasis*) or after learning is completed (*Proj-Oasis*). Alternatively, the asymmetric score function $S_{\mathbf{W}}(p_i, p_j)$ in $l_W$ can be replaced with a symmetric score

$$S'_{\mathbf{W}}(p_i, p_j) \equiv -(p_i - p_j)^T \, \mathbf{W} \, (p_i - p_j) \, . \tag{6}$$

and used to derive an OASIS-like algorithm (which we name *Dissim-Oasis*). The optimal update for this loss has a symmetric gradient $\mathbf{V}'^i = (p_i - p_i^+)(p_i - p_i^+)^T - (p_i - p_i^-)(p_i - p_i^-)^T$. Therefore, if $\mathbf{W}^0$ is initialized with a symmetric matrix (e.g., the identity) all $\mathbf{W}^i$ are guaranteed to remain

symmetric. *Dissim-Oasis* is closely related to LMNN [16]. This can be seen be casting the batch objective of LMNN, into an online setup, which has the form $err(W) = -\omega \cdot S'_{\mathbf{W}}(p_i, p_i^+) + (1 - \omega) \cdot l'_{\mathbf{W}}(p_i, p_i^+, p_i^-)$. This online version of LMNN becomes equivalent to *D*issim-Oasis for $\omega = 0$.

Figure 3(**a**) compares the precision of the different symmetric variants with the original OA-SIS. All symmetric variants performed slightly worse, or equal, to the original asymmetric OA-SIS. The precision of *Proj-Oasis* was equivalent to that of OASIS, most likely since asymmetric OASIS actually converged to an almost-symmetric model (as measured by a symmetry index $\rho(\mathbf{W}) = \frac{\|sym(\mathbf{W})\|_2}{\|\mathbf{W}\|_2} = 0.94$).

### 4.3.2 Positive similarity

Most similarity learning approaches focus on learning metrics. In the context of OASIS, when $\mathbf{W}$ is positive semi definite (PSD), it defines a Mahalanobis distance over the images. The matrix square-root of $\mathbf{W}$, $\mathbf{A}^T\mathbf{A} = \mathbf{W}$ can then be used to project the data into a new space in which the Euclidean distance is equivalent to the $\mathbf{W}$ distance in the original space.

We experimented with positive variants of OASIS, where we repeatedly projected the learned model onto the set of PSD matrices, once every $t$ iterations. Projection is done by taking the eigen decomposition $\mathbf{W} = \mathbf{V} \cdot \mathbf{D} \cdot \mathbf{V}^T$ where $\mathbf{V}$ is the eigenvector matrix and $\mathbf{D}$ is a the diagonal eigenvalues matrix limited to positive eigenvalues. Figure 3(b) traces precision on the test set throughout learning for various values of $t$.

The effect of positive projections is complex. First, continuously projecting at every step helps to reduce overfitting, as can be observed by the slower decline of the blue curve (upper smooth curve) compared to the orange curve (lowest curve). However, when projection is performed after many steps, (instead of continuously), performance of the projected model actually outperforms the continuous-projection model (upper jittery curve). The reason for this effect is likely to be that estimating the positive sub-space is very noisy when only based on a few samples. Indeed, accurate estimation of the negative subspace is known to be a hard problem, in that the estimated eigenvalues of eigenvectors "near zero", is relatively large. We found that this effect was so strong, that the optimal projection strategy is to avoid projection throughout learning completely. Instead, projecting into PSD after learning (namely, after a model was chosen using early stopping) provided the best performance in our experiments.

An interesting alternative to obtain a PSD matrix was explored by [10, 9]. Using a LogDet divergence between two matrices $D_{ld}(X,Y) = tr(XY^{-1}) - log(det(XY^{-1}))$ ensures that, given an initial PSD matrix, all subsequent matrices will be PSD as well. It will be interesting to test the effect of using LogDet regularization in the OASIS setup.

## 5  Discussion

We have presented OASIS, a scalable algorithm for learning image similarity that captures both semantic and visual aspects of image similarity. Three key factors contribute to the scalability of OASIS. First, using a large margin online approach allows training to converge even after seeing a small fraction of potential pairs. Second, the objective function of OASIS does not require the similarity measure to be a metric during training, although it appears to converge to a near-symmetric solution, whose positive projection is a good metric. Finally, we use a sparse representation of low level features which allows to compute scores very efficiently.

OASIS learns a class-independent model: it is not aware of which queries or categories were shared by two similar images. As such, it is more limited in its descriptive power and it is likely that class-dependent similarity models could improve precision. On the other hand, class-independent models could generalize to handle classes that were not observed during training, as in transfer learning. Large scale similarity learning, applied to images from a large variety of classes, could therefore be a useful tool to address real-world problems with a large number of classes.

This paper focused on the training part of metric learning. To use the learned metric for ranking, an efficient procedure for scoring a large set of images is needed. Techniques based on locality-sensitive hashing could be used to speed up evaluation, but this is outside the scope of this paper.

## Footnotes

[1]We have also experimented with the methods of [18], which we found to be too slow, and with RCA [1], whose precision was lower than other methods. These results are not included in the evaluations below.

# References

[1] A. Bar-Hillel, T. Hertz, N. Shental, and D. Weinshall. Learning Distance Functions using Equivalence Relations. In *Proc. of 20th International Conference on Machine Learning (ICML)*, pages 11–18, 2003.

[2] K. Crammer, O. Dekel, J. Keshet, S. Shalev-Shwartz, and Y. Singer. Online passive-aggressive algorithms. *JMLR*, 7:551–585, 2006.

[3] J.V. Davis, B. Kulis, P. Jain, S. Sra, and I.S. Dhillon. Information-theoretic metric learning. In *ICML 24*, pages 209–216, 2007.

[4] A. Frome, Y. Singer, F. Sha, and J. Malik. Learning globally-consistent local distance functions for shape-based image retrieval and classification. In *International Conference on Computer Vision*, pages 1–8, 2007.

[5] A. Globerson and S. Roweis. Metric Learning by Collapsing Classes. *NIPS*, 18:451, 2006.

[6] D. Grangier and S. Bengio. A discriminative kernel-based model to rank images from text queries. *Transactions on Pattern Analysis and Machine Intelligence (TPAMI)*, 30(8):1371–1384, 2008.

[7] G. Griffin, A. Holub, and P. Perona. Caltech-256 object category dataset. Technical Report 7694, CalTech, 2007.

[8] R. Hadsell, S. Chopra, and Y. LeCun. Dimensionality reduction by learning an invariant mapping. In *IEEE Computer Society Conference on Computer Vision and Pattern Recognition (CVPR)*, volume 2, 2006.

[9] P. Jain, B. Kulis, I. Dhillon, and K. Grauman. Online metric learning and fast similarity search. In *NIPS*, volume 22, 2008.

[10] B. Kulis, M.A. Sustik, and I.S. Dhillon. Low-rank kernel learning with bregman matrix divergences. *Journal of Machine Learning Research*, 10:341–376, 2009.

[11] G.R.G. Lanckriet, N. Cristianini, P. Bartlett, L. El Ghaoui, and M.I. Jordan. Learning the kernel matrix with semidefinite programming. *JMLR*, 5:27–72, 2004.

[12] W. S. Noble. Multi-kernel learning for biology. In *NIPS workshop on kernel learning*, 2008.

[13] N. Rasiwasia and N. Vasconcelos. A study of query by semantic example. In *3rd International Workshop on Semantic Learning and Applications in Multimedia*, 2008.

[14] R. Rosales and G. Fung. Learning sparse metrics via linear programming. In *Proceedings of the 12th ACM SIGKDD international conference on Knowledge discovery and data mining*, pages 367–373. ACM New York, NY, USA, 2006.

[15] A. Tversky. Features of similarity. *Psychological Review*, 84(4):327–352, 1977.

[16] K. Weinberger, J. Blitzer, and L. Saul. Distance metric learning for large margin nearest neighbor classification. *NIPS*, 18:1473, 2006.

[17] K.Q. Weinberger and L.K. Saul. Fast solvers and efficient implementations for distance metric learning. In *ICML25*, pages 1160–1167, 2008.

[18] E.P. Xing, A.Y. Ng, M.I. Jordan, and S. Russell. Distance metric learning with application to clustering with side-information. In S. Becker, S. Thrun, and K. Obermayer, editors, *NIPS 15*, pages 521–528, Cambridge, MA, 2003. MIT Press.

[19] L. Yang. Distance metric learning: A comprehensive survey. Technical report, Michigan State Univ., 2006.

